# A PAC-Bayesian Margin Bound for Linear Classifiers: Why SVMs work

**Ralf Herbrich**
Statistics Research Group
Computer Science Department
Technical University of Berlin
*ralfh@cs.tu-berlin.de*

**Thore Graepel**
Statistics Research Group
Computer Science Department
Technical University of Berlin
*guru@cs.tu-berlin.de*

## Abstract

We present a bound on the generalisation error of linear classifiers in terms of a refined margin quantity on the training set. The result is obtained in a PAC–Bayesian framework and is based on geometrical arguments in the space of linear classifiers. The new bound constitutes an exponential improvement of the so far tightest margin bound by Shawe-Taylor et al. [8] and scales logarithmically in the inverse margin. Even in the case of less training examples than input dimensions sufficiently large margins lead to non-trivial bound values and — for maximum margins — to a vanishing complexity term. Furthermore, the classical margin is too coarse a measure for the *essential quantity* that controls the generalisation error: the *volume ratio* between the whole hypothesis space and the subset of consistent hypotheses. The practical relevance of the result lies in the fact that the well-known support vector machine is optimal w.r.t. the new bound only if the feature vectors are all of the same length. As a consequence we recommend to use SVMs on normalised feature vectors only — a recommendation that is well supported by our numerical experiments on two benchmark data sets.

## 1 Introduction

Linear classifiers are exceedingly popular in the machine learning community due to their straight-forward applicability and high flexibility which has recently been boosted by the so-called kernel methods [13]. A natural and popular framework for the theoretical analysis of classifiers is the PAC (*probably approximately correct*) framework [11] which is closely related to Vapnik's work on the generalisation error [12]. For binary classifiers it turned out that the *growth function* is an appropriate measure of "complexity" and can tightly be upper bounded by the VC (Vapnik-Chervonenkis) dimension [14]. Later, *structural risk minimisation* [12] was suggested for directly minimising the VC dimension based on a training set and an *a priori* structuring of the hypothesis space.

In practice, e.g. in the case of linear classifiers, often a thresholded *real-valued* func-

tion is used for classification. In 1993, Kearns [4] demonstrated that considerably tighter bounds can be obtained by considering a scale-sensitive complexity measure known as the *fat shattering dimension*. Further results [1] provided bounds on the Growth function similar to those proved by Vapnik and others [14, 6]. The popularity of the theory was boosted by the invention of the *support vector machine* (SVM) [13] which aims at directly minimising the complexity as suggested by theory.

Until recently, however, the success of the SVM remained somewhat obscure because in PAC/VC theory the structuring of the hypothesis space must be *independent* of the training data — in contrast to the data-dependence of the canonical hyperplane. As a consequence Shawe-Taylor et.al. [8] developed the *luckiness framework*, where luckiness refers to a complexity measure that is a function of both hypothesis *and* training sample.

Recently, David McAllester presented some PAC-Bayesian theorems [5] that bound the generalisation error of Bayesian classifiers *independently* of the correctness of the prior and regardless of the underlying data distribution — thus fulfilling the basic desiderata of PAC theory. In [3] McAllester's bounds on the Gibbs classifier were extended to the Bayes (optimal) classifier. The PAC-Bayesian framework provides *a posteriori* bounds and is thus closely related in spirit to the luckiness framework[1].

In this paper we give a tight margin bound for linear classifiers in the PAC-Bayesian framework. The main idea is to identify the generalisation error of the classifier $h$ of interest with that of the Bayes (optimal) classifier of a (point-symmetric) subset $Q$ that is *summarised* by $h$. We show that for a uniform prior the *normalised margin* of $h$ is *directly* related to the volume of a large subset $Q$ summarised by $h$. In particular, the result suggests that a learning algorithm for linear classifiers should aim at maximising the normalised margin instead of the classical margin. In Section 2 and 3 we review the basic PAC-Bayesian theorem and show how it can be applied to single classifiers. In Section 4 we give our main result and outline its proof. In Section 5 we discuss the consequences of the new result for the application of SVMs and demonstrate experimentally that in fact a normalisation of the feature vectors leads to considerably superior generalisation performance.

We denote $n$–tuples by italic bold letters (e.g. $\boldsymbol{x} = (x_1, \ldots, x_n)$), vectors by roman bold letters (e.g. $\mathbf{x}$), random variables by sans serif font (e.g. $\mathsf{X}$) and vector spaces by calligraphic capitalised letters (e.g. $\mathcal{X}$). The symbols $\mathsf{P}, \mathsf{E}, \mathsf{I}$ and $\ell_2^n$ denote a probability measure, the expectation of a random variable, the indicator function and the normed space (2–norm) of sequences of length $n$, respectively.

## 2  A PAC Margin Bound

We consider learning in the PAC framework. Let $\mathcal{X}$ be the input space, and let $\mathcal{Y} = \{-1, +1\}$. Let a labelled training sample $\boldsymbol{z} = (\boldsymbol{x}, \boldsymbol{y}) \in (\mathcal{X} \times \mathcal{Y})^m = \mathcal{Z}^m$ be drawn iid according to some unknown probability measure $\mathsf{P}_\mathsf{Z} = \mathsf{P}_{\mathsf{Y}|\mathsf{X}}\mathsf{P}_\mathsf{X}$. Furthermore for a given hypothesis space $\mathcal{H} \subseteq \mathcal{Y}^{\mathcal{X}}$ we assume the existence of a "true" hypothesis $h^* \in \mathcal{H}$ that labelled the data

$$\mathsf{P}_{\mathsf{Y}|\mathsf{X}=x}(y) = \mathsf{I}_{y=h^*(x)}. \tag{1}$$

We consider linear hypotheses

$$\mathcal{H} = \{ h_\mathbf{w} : x \mapsto \text{sign}\left(\langle \mathbf{w}, \boldsymbol{\phi}(x) \rangle_\mathcal{K}\right) \mid \mathbf{w} \in \mathcal{W} \}, \quad \mathcal{W} = \{ \mathbf{w} \in \mathcal{K} \mid \|\mathbf{w}\|_\mathcal{K} = 1 \}, \tag{2}$$

where the mapping $\phi : \mathcal{X} \to \mathcal{K} \subseteq \ell_2^n$ maps[2] the input data to some feature space $\mathcal{K}$ and $\|\mathbf{w}\|_{\mathcal{K}} = 1$ leads to a one-to-one correspondence of hypotheses $h_{\mathbf{w}}$ to their parameters $\mathbf{w}$. From the existence of $h^*$ we know that there exists a version space $V(z) \subseteq \mathcal{W}$,

$$V(z) = \{\mathbf{w} \in \mathcal{W} \mid \forall (x, y) \in z \, : \, h_{\mathbf{w}}(x) = y\} \, .$$

Our analysis aims at bounding the true risk $R[\mathbf{w}]$ of consistent hypotheses $h_{\mathbf{w}}$,

$$R[\mathbf{w}] = \mathbf{P}_{\mathsf{XY}}(h_{\mathbf{w}}(\mathsf{X}) \neq \mathsf{Y}) \, .$$

Since all classifiers $\mathbf{w} \in V(z)$ are indistinguishable in terms of number of errors committed on the given training set $z$ let us introduce the concept of the *margin* $\gamma_z(\mathbf{w})$ of a classifier $\mathbf{w}$, i.e.

$$\gamma_z(\mathbf{w}) = \min_{(x_i, y_i) \in z} \frac{y_i \langle \mathbf{w}, \mathbf{x}_i \rangle_{\mathcal{K}}}{\|\mathbf{w}\|_{\mathcal{K}}} \, . \tag{3}$$

The following theorem due to Shawe-Taylor et al. [8] bounds the generalisation errors $R[\mathbf{w}]$ of all classifier $\mathbf{w} \in V(z)$ in terms of the margin $\gamma_z(\mathbf{w})$.

**Theorem 1 (PAC margin bound).** *For all probability measures $\mathbf{P}_{\mathsf{Z}}$ such that $\mathbf{P}_{\mathsf{X}}(\|\phi(\mathsf{X})\|_{\mathcal{K}} \leq \varsigma) = 1$, for any $\delta > 0$ with probability at least $1 - \delta$ over the random draw of the training set $z$, if we succeed in correctly classifying $m$ samples $z$ with a linear classifier $\mathbf{w}$ achieving a positive margin $\gamma_z(\mathbf{w}) > \sqrt{32\varsigma^2/m}$ then the generalisation $R[\mathbf{w}]$ of $\mathbf{w}$ is bounded from above by*

$$\frac{2}{m}\left(\kappa(\mathbf{w}) \log_2\left(\frac{8em}{\kappa(\mathbf{w})}\right) \log_2(32m) + \ln\left(\frac{m^2}{\delta}\right)\right), \quad \kappa(\mathbf{w}) = \left\lceil \left(\frac{8\varsigma}{\gamma_z(\mathbf{w})}\right)^2 \right\rceil . \tag{4}$$

As the bound on $R[\mathbf{w}]$ depends linearly on $\gamma_z^{-2}(\mathbf{w})$ we see that Theorem 1 provides a theoretical foundation of all algorithms that aim at maximising $\gamma_z(\mathbf{w})$, e.g. SVMs and Boosting [13, 7].

## 3 PAC-Bayesian Analysis

We first present a result [5] that bounds the risk of the generalised Gibbs classification strategy $Gibbs_{W(z)}$ by the measure $\mathbf{P}_{\mathsf{W}}(W(z))$ on a consistent subset $W(z) \subseteq V(z)$. This average risk is then related via the Bayes-Gibbs lemma to the risk of the Bayes classification strategy $Bayes_{W(z)}$ on $W(z)$. For a single consistent hypothesis $\mathbf{w} \in \mathcal{W}$ it is then necessary to identify a consistent subset $Q(\mathbf{w})$ such that the Bayes strategy $Bayes_{Q(\mathbf{w})}$ on $Q(\mathbf{w})$ always agrees with $\mathbf{w}$. Let us define the Gibbs classification strategy $Gibbs_{W(z)}$ w.r.t. the subset $W(z) \subseteq V(z)$ by

$$Gibbs_{W(z)}(x) = h_{\mathbf{w}}(x) \, , \qquad \mathbf{w} \sim \mathbf{P}_{\mathsf{W}|\mathsf{W} \in W(z)} \, . \tag{5}$$

Then the following theorem [5] holds for the risk of $Gibbs_{W(z)}$.

**Theorem 2 (PAC-Bayesian bound for subsets of classifiers).** *For any measure $\mathbf{P}_{\mathsf{W}}$ and any measure $\mathbf{P}_{\mathsf{Z}}$, for any $\delta > 0$ with probability at least $1 - \delta$ over the random draw of the training set $z$ for all subsets $W(z) \subseteq V(z)$ such that $\mathbf{P}_{\mathsf{W}}(W(z)) > 0$ the generalisation error of the associated Gibbs classification strategy $Gibbs_{W(z)}$ is bounded from above by*

$$R[Gibbs_{W(z)}] \leq \frac{1}{m}\left(\ln\left(\frac{1}{\mathbf{P}_{\mathsf{W}}(W(z))}\right) + 2\ln(m) + \ln\left(\frac{1}{\delta}\right) + 1\right) . \tag{6}$$

Now consider the Bayes classifier $Bayes_{W(z)}$,

$$Bayes_{W(z)}(x) = \text{sign} \left( \mathbf{E}_{\mathbf{W}|\mathbf{W} \in W(z)} \left[ h_{\mathbf{W}}(x) \right] \right),$$

where the expectation $\mathbf{E}_{\mathbf{W}|\mathbf{W} \in W(z)}$ is taken over a cut-off posterior given by combining the PAC-likelihood (1) and the prior $\mathbf{P_W}$.

**Lemma 1 (Bayes-Gibbs Lemma).** *For any two measures* $\mathbf{P_W}$ *and* $\mathbf{P_{XY}}$ *and any set* $W \subseteq \mathcal{W}$

$$\mathbf{P_{XY}} \left( Bayes_W(\mathsf{X}) \neq \mathsf{Y} \right) \leq 2 \cdot \mathbf{P_{XY}} \left( Gibbs_W(\mathsf{X}) \neq \mathsf{Y} \right). \tag{7}$$

*Proof. (Sketch)* Consider only the simple PAC setting we need. At all those points $x \in \mathcal{X}$ at which $Bayes_W$ is wrong by definition at least half of the classifiers $\mathbf{w} \in W$ under consideration make a mistake as well. $\quad\square$

The combination of Lemma 1 with Theorem 2 yields a bound on the risk of $Bayes_{W(z)}$. For a single hypothesis $\mathbf{w} \in \mathcal{W}$ let us find a (Bayes-admissible) subset $Q(\mathbf{w})$ of version space $V(z)$ such that $Bayes_{Q(\mathbf{w})}$ on $Q(\mathbf{w})$ agrees with $\mathbf{w}$ on every point in $\mathcal{X}$.

**Definition 1 (Bayes-admissibility).** Given the hypothesis space in (2) and a prior measure $\mathbf{P_W}$ over $\mathcal{W}$ we call a subset $Q(\mathbf{w}) \subseteq \mathcal{W}$ *Bayes admissible w.r.t.* $\mathbf{w}$ *and* $\mathbf{P_W}$ if and only if

$$\forall x \in \mathcal{X} : \qquad h_{\mathbf{w}}(x) = Bayes_{Q(\mathbf{w})}(x).$$

Although difficult to achieve in general the following geometrically plausible lemma establishes Bayes-admissibility for the case of interest.

**Lemma 2 (Bayes-admissibility for linear classifiers).** *For uniform measure* $\mathbf{P_W}$ *over* $\mathcal{W}$ *each ball* $Q(\mathbf{w}) = \{ \mathbf{v} \in \mathcal{W} \mid \|\mathbf{w} - \mathbf{v}\|_{\mathcal{K}} \leq r \}$ *is Bayes admissible w.r.t. its centre* $\mathbf{w}$.

Please note that by considering a ball $Q(\mathbf{w})$ rather than just $\mathbf{w}$ we make use of the fact that $\mathbf{w}$ *summarises* all its neighbouring classifiers $\mathbf{v} \in Q(\mathbf{w})$. Now using a uniform prior $\mathbf{P_W}$ the *normalised* margin

$$\Gamma_z(\mathbf{w}) = \min_{(x_i, y_i) \in z} \frac{y_i \langle \mathbf{w}, \mathbf{x}_i \rangle_{\mathcal{K}}}{\|\mathbf{w}\|_{\mathcal{K}} \|\mathbf{x}_i\|_{\mathcal{K}}}, \tag{8}$$

quantifies the relative volume of classifiers summarised by $\mathbf{w}$ and thus allows us to bound its risk. Note that in contrast to the classical margin $\gamma_z$ (see 3) this *normalised* margin is a dimensionless quantity and constitutes a measure for the relative size of the version space invariant under rescaling of both weight vectors $\mathbf{w}$ and feature vectors $\mathbf{x}_i$.

## 4  A PAC-Bayesian Margin Bound

Combining the ideas outlined in the previous section allows us to derive a generalisation error bound for linear classifiers $\mathbf{w} \in V(z)$ in terms of their *normalised* margin $\Gamma_z(\mathbf{w})$.

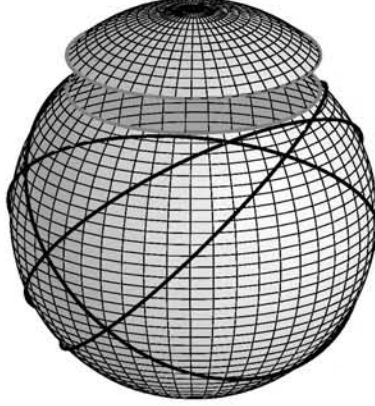

Figure 1: Illustration of the volume ratio for the classifier at the north pole. Four training points shown as grand circles make up version space — the polyhedron on top of the sphere. The radius of the "cap" of the sphere is proportional to the margin $\Gamma_{\boldsymbol{z}}$, which *only* for constant $\|\mathbf{x}_i\|_{\mathcal{K}}$ is maximised by the SVM.

---

**Theorem 3 (PAC-Bayesian margin bound).** *Suppose $\mathcal{K} \subseteq \ell_2^n$ is a given feature space of dimensionality $n$. For all probability measures $\mathbf{P}_Z$, for any $\delta > 0$ with probability at least $1 - \delta$ over the random draw of the training set $\boldsymbol{z}$, if we succeed in correctly classifying $m$ samples $\boldsymbol{z}$ with a linear classifier $\mathbf{w}$ achieving a positive margin $\Gamma_{\boldsymbol{z}}(\mathbf{w}) > 0$ then the generalisation error $R[\mathbf{w}]$ of $\mathbf{w}$ is bounded from above by*

$$\frac{2}{m} \left( d \ln \left( \frac{1}{1 - \sqrt{1 - \Gamma_{\boldsymbol{z}}^2(\mathbf{w})}} \right) + 2 \ln(m) + \ln \left( \frac{1}{\delta} \right) + 2 \right). \tag{9}$$

*where $d = \min(m, n)$.*

---

*Proof.* Geometrically the hypothesis space $\mathcal{W}$ is the unit sphere in $\mathbb{R}^n$ (see Figure 1). Let us assume that $\mathbf{P}_{\mathbf{W}}$ is uniform on the unit sphere as suggested by symmetry. Given the training set $\boldsymbol{z}$ and a classifier $\mathbf{w}$ all classifiers $\mathbf{v} \in Q(\mathbf{w})$

$$Q(\mathbf{w}) = \left\{ \mathbf{v} \in \mathcal{W} \mid \langle \mathbf{w}, \mathbf{v} \rangle_{\mathcal{K}} > \sqrt{1 - \Gamma_{\boldsymbol{z}}^2(\mathbf{w})} \right\} \tag{10}$$

are within $V(\boldsymbol{z})$ (For a proof see [2]). Such a set $Q(\mathbf{w})$ is Bayes-admissible by Lemma 2 and hence we can use $\mathbf{P}_{\mathbf{W}}(Q(\mathbf{w}))$ to bound the generalisation error of $\mathbf{w}$. Since $\mathbf{P}_{\mathbf{W}}$ is uniform, the value $-\ln(\mathbf{P}_{\mathbf{W}}(Q(\mathbf{w})))$ is simply the logarithm of the *volume ratio* between the surface of the unit sphere and the surface of all $\mathbf{v}$ fulfilling equation (10). In [2] it is shown that this ratio is *exactly* given by

$$\ln \left( \frac{\int_0^{2\pi} \sin^{n-2}(\theta) \, d\theta}{\int_0^{\arccos\left(\sqrt{1 - \Gamma_{\boldsymbol{z}}^2(\mathbf{w})}\right)} \sin^{n-2}(\theta) \, d\theta} \right).$$

It can be shown that this ratio is tightly bounded from above by

$$n \ln \left( \frac{1}{1 - \sqrt{1 - \Gamma_{\boldsymbol{z}}^2(\mathbf{w})}} \right) + \ln(2).$$

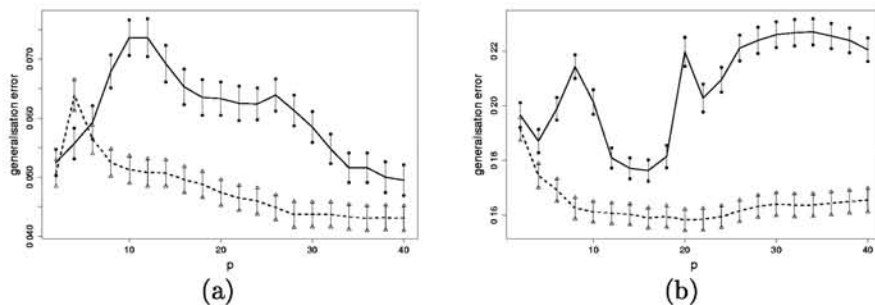

Figure 2: Generalisation errors of classifiers learned by an SVM with (dashed line) and without (solid line) normalisation of the feature vectors $\mathbf{x}_i$. The error bars indicate one standard deviation over 100 random splits of the data sets. The two plots are obtained on the **(a)** `thyroid` and **(b)** `sonar` data set.

With $\ln(2) < 1$ we obtain the desired result. Note that $m$ points maximally span an $m$–dimensional space and thus we can marginalise over the remaining $n - m$ dimensions of feature space $\mathcal{K}$. This gives $d = \min(m, n)$. $\qquad\square$

An appealing feature of equation (9) is that for $\Gamma_{\mathbf{z}}(\mathbf{w}) = 1$ the bound reduces to $\frac{2}{m}(2\ln(m) - \ln(\delta) + 2)$ with a rapid decay to zero as $m$ increases. In case of margins $\Gamma_{\mathbf{z}}(\mathbf{w}) > 0.91$ the troublesome situation of $d = m$, which occurs e.g. for RBF kernels, is compensated for. Furthermore, upper bounding $1/(1 - \sqrt{1 - \Gamma})$ by $2/\Gamma$ we see that Theorem 3 is an exponential improvement of Theorem 1 in terms of the attained margins. It should be noted, however, that the new bound depends on the dimensionality of the input space via $d = \min(m, n)$.

## 5 Experimental Study

Theorem 3 suggest the following learning algorithm: given a version space $V(\mathbf{z})$ (through a given training set $\mathbf{z}$) find the classifier $\mathbf{w}$ that maximises $\Gamma_{\mathbf{z}}(\mathbf{w})$. This algorithm, however, is given by the SVM *only if* the training data in feature space $\mathcal{K}$ are normalised. We investigate the influence of such a normalisation on the generalisation error in the feature space $\mathcal{K}$ of all monomials up to the $p$–th degree (well-known from handwritten digit recognition, see [13]). Since the SVM learning algorithm as well as the resulting classifier only refer to inner products in $\mathcal{K}$, it suffices to use an easy-to-calculate kernel function $k : \mathcal{X} \times \mathcal{X} \to \mathbb{R}$ such that for all $x, x' \in \mathcal{X}$, $k(x, x') = \langle \phi(x), \phi(x') \rangle_{\mathcal{K}}$, given in our case by the polynomial kernel

$$\forall p \in \mathbb{N}: \qquad k(x, x') = (\langle x, x' \rangle_{\mathcal{X}} + 1)^p .$$

Earlier experiment have shown [13] that without normalisation too large values of $p$ may lead to "overfitting". We used the UCI [10] data sets `thyroid` ($d = 5$, $m = 140$, $m_{\text{test}} = 75$) and `sonar` ($d = 60$, $m = 124$, $m_{\text{test}} = 60$) and plotted the generalisation error of SVM solutions (estimated over 100 different splits of the data set) as a function of $p$ (see Figure 2). As suggested by Theorem 3 in almost all cases the normalisation improved the performance of the support vector machine solution at a statistically significant level. As a consequence, we recommend:

> When training an SVM, always normalise your data in feature space.

Intuitively, it is only the *spatial direction* of both weight vector and feature vectors that determines the classification. Hence the different lengths of feature vectors in the training set should not enter the SVM optimisation problem.

## 6 Conclusion

The PAC-Bayesian framework together with simple geometrical arguments yields the so far tightest margin bound for linear classifiers. The role of the normalised margin $\Gamma_z$ in the new bound suggests that the SVM is theoretically justified only for input vectors of constant length. We hope that this result is recognised as a useful bridge between theory and practice in the spirit of Vapnik's famous statement:

> Nothing is more practical than a good theory

**Acknowledgements** We would like to thank David McAllester, John Shawe-Taylor, Bob Williamson, Olivier Chapelle, John Langford, Alex Smola and Bernhard Schölkopf for interesting discussions and useful suggestions on earlier drafts.

## Footnotes

[1]In fact, even Shawe-Taylor et.al. concede that "... a Bayesian might say that luckiness is just a complicated way of encoding a prior. The sole justification for our particular way of encoding is that it allows us to get the PAC like results we sought..." [9, p. 4].

[2]For notational simplicity we sometimes abbreviate $\phi(x)$ by $\mathbf{x}$ which should not be confused with the sample $x$ of training objects.

## References

[1] N. Alon, S. Ben-David, N. Cesa-Bianchi, and D. Haussler. Scale sensitive dimensions, uniform convergence and learnability. *Journal of the ACM*, 44(4):615–631, 1997.

[2] R. Herbrich. *Learning Linear Classifiers - Theory and Algorithms*. PhD thesis, Technische Universität Berlin, 2000. accepted for publication by MIT Press.

[3] R. Herbrich, T. Graepel, and C. Campbell. Bayesian learning in reproducing kernel Hilbert spaces. Technical report, Technical University of Berlin, 1999. TR 99-11.

[4] M. J. Kearns and R. Schapire. Efficient distribution-free learning of probabilistic concepts. *Journal of Computer and System Sciences*, 48(2):464–497, 1993.

[5] D. A. McAllester. Some PAC Bayesian theorems. In *Proceedings of the Eleventh Annual Conference on Computational Learning Theory*, pages 230–234, Madison, Wisconsin, 1998.

[6] N. Sauer. On the density of families of sets. *Journal of Combinatorial Theory, Series A*, 13:145–147, 1972.

[7] R. E. Schapire, Y. Freund, P. Bartlett, and W. S. Lee. Boosting the margin: A new explanation for the effectiveness of voting methods. In *Proceedings of the 14-th International Conference in Machine Learning*, 1997.

[8] J. Shawe-Taylor, P. L. Bartlett, R. C. Williamson, and M. Anthony. Structural risk minimization over data–dependent hierarchies. *IEEE Transactions on Information Theory*, 44(5):1926–1940, 1998.

[9] J. Shawe-Taylor and R. C. Williamson. A PAC analysis of a Bayesian estimator. Technical report, Royal Holloway, University of London, 1997. NC2–TR–1997–013.

[10] UCI. University of California Irvine: Machine Learning Repository, 1990.

[11] L. G. Valiant. A theory of the learnable. *Communications of the ACM*, 27(11):1134–1142, 1984.

[12] V. Vapnik. *Estimation of Dependences Based on Empirical Data*. Springer, 1982.

[13] V. Vapnik. *The Nature of Statistical Learning Theory*. Springer, 1995.

[14] V. Vapnik and A. Chervonenkis. On the uniform convergence of relative frequencies of events to their probabilities. *Theory of Probability and its Application*, 16(2):264–281, 1971.